# Greedy Model Averaging

**Dong Dai**
Department of Statistics Rutgers University, New Jersey, 08816
dongdai916@gmail.com

**Tong Zhang**
Department of Statistics, Rutgers University, New Jersey, 08816
tzhang@stat.rutgers.edu

## Abstract

This paper considers the problem of combining multiple models to achieve a prediction accuracy not much worse than that of the best single model for least squares regression. It is known that if the models are mis-specified, model averaging is superior to model selection. Specifically, let $n$ be the sample size, then the worst case regret of the former decays at the rate of $O(1/n)$ while the worst case regret of the latter decays at the rate of $O(1/\sqrt{n})$. In the literature, the most important and widely studied model averaging method that achieves the optimal $O(1/n)$ average regret is the exponential weighted model averaging (EWMA) algorithm. However this method suffers from several limitations. The purpose of this paper is to present a new greedy model averaging procedure that improves EWMA. We prove strong theoretical guarantees for the new procedure and illustrate our theoretical results with empirical examples.

## 1 Introduction

This paper considers the model combination problem, where the goal is to combine multiple models in order to achieve improved accuracy. This problem is important for practical applications because it is often the case that single learning models do not perform as well as their combinations. In practice, model combination is often achieved through the so-called "stacking" procedure, where multiple models $\{f_1(x), \ldots, f_M(x)\}$ are first learned based on a shared "training dataset". Then these models are combined on a separate "validation dataset". This paper is motivated by this scenario. In particular, we assume that $M$ models $\{f_1(x), \ldots, f_M(x)\}$ are given a priori (e.g., we may regard them as being obtained with a separate training set), and we are provided with $n$ labeled data points (validation data) $\{(X_1, Y_1), \ldots, (X_n, Y_n)\}$ to combine these models.

For simplicity and clarity, our analysis focuses on least squares regression in fixed design although similar analysis can be extended to random design and to other loss functions. In this setting, for notation convenience, we can represent the $k$-th model on the validation data as a vector $\boldsymbol{f}_k = [f_k(X_1), \ldots, f_k(X_n)] \in \mathbb{R}^n$, and we let the observation vector $\boldsymbol{y} = [Y_1, \ldots, Y_n] \in \mathbb{R}^n$. Let $\boldsymbol{g} = \mathbb{E}\boldsymbol{y}$ be the mean. Our goal (in the fixed design or denoising setting) is to estimate the mean vector $\boldsymbol{g}$ from $\boldsymbol{y}$ using the $M$ existing models $\mathcal{F} = \{\boldsymbol{f}_1, \ldots \boldsymbol{f}_M\}$. Here, we can write

$$\boldsymbol{y} = \boldsymbol{g} + \boldsymbol{\xi},$$

where we assume that $\boldsymbol{\xi}$ are iid Gaussian noise: $\boldsymbol{\xi} \sim N(0, \sigma^2 \boldsymbol{I}_{n \times n})$ for simplicity. This iid Gaussian assumption isn't critical, and the results remain the same for independent sub-Gaussian noise.

We assume that the models may be mis-specified. That is, let $k_*$ be the best single model defined as:

$$k_* = \underset{k}{\operatorname{argmin}} \, \|\boldsymbol{f}_k - \boldsymbol{g}\|_2^2, \tag{1}$$

then $\boldsymbol{f}_{k_*} \neq \boldsymbol{g}$.

We are interested in an estimator $\hat{\boldsymbol{f}}$ of $\boldsymbol{g}$ that achieves a small regret

$$R(\hat{\boldsymbol{f}}) = \frac{1}{n} \left\| \hat{\boldsymbol{f}} - \boldsymbol{g} \right\|_2^2 - \frac{1}{n} \left\| \boldsymbol{f}_{k_*} - \boldsymbol{g} \right\|_2^2 .$$

This paper considers a special class of model combination methods which we refer to as *model averaging*, with combined estimators of the form

$$\hat{\boldsymbol{f}} = \sum_{k=1}^{M} \hat{w}_k \boldsymbol{f}_k,$$

where $\hat{w}_k \geq 0$ and $\sum_k \hat{w}_k = 1$. A standard method for "model averaging" is model selection, where we choose the model $\hat{k}$ with the smallest least squares error:

$$\hat{\boldsymbol{f}}_{MS} = \boldsymbol{f}_{\hat{k}}; \qquad \hat{k} = \arg\min_k \|\boldsymbol{f}_k - \boldsymbol{y}\|_2^2 .$$

This corresponds to the choice of $\hat{w}_{\hat{k}} = 1$ and $\hat{w}_k = 0$ when $k \neq \hat{k}$. However, it is well known that the worst case regret this procedure can achieve is $R(\hat{\boldsymbol{f}}_{MS}) = O(\sqrt{\ln M/n})$ [1]. Another standard model averaging method is the Exponential Weighted Model Averaging (EWMA) estimator defined as

$$\hat{\boldsymbol{f}}_{EWMA} = \sum_{k=1}^{M} \hat{w}_k \boldsymbol{f}_k, \quad \hat{w}_k = \frac{q_k e^{-\lambda \|\boldsymbol{f}_k - \boldsymbol{y}\|_2^2}}{\sum_{j=1}^{M} q_j e^{-\lambda \|\boldsymbol{f}_j - \boldsymbol{y}\|_2^2}}, \tag{2}$$

with a tuned parameter $\lambda \geq 0$. The extra parameters $\{q_j\}_{j=1,\dots,M}$ are priors that impose bias favoring some models over some other models. Here we assume that $q_j \geq 0$ and $\sum_j q_j = 1$. In this setting, the most common prior choice is the flat prior $q_j = 1/M$. It should be pointed out that a progressive variant of (2), which returns the average of $n + 1$ EWMA estimators with $S_i = \{(X_1, Y_1), \dots, (X_i, Y_i)\}$ for $i = 0, 1, \dots, n$, was often analyzed in the earlier literature [2, 9, 5, 1]. Nevertheless, the non progressive version presented in (2) is clearly a more natural estimator, and this is the form that has been studied in more recent work [3, 6, 8]. Our current paper does not differentiate these two versions of EWMA because they have similar theoretical properties. In particular, our experiments only compare to the non-progressive version (2) that performs better in practice.

It is known that exponential model averaging leads to an average regret of $O(\ln M/n)$ which achieves the optimal rate; however it was pointed out in [1] that the rate does not hold with large probability. Specifically, EWMA only leads to a sub-optimal deviation bound of $O(\sqrt{\ln M/n})$ with large probability. To remedy this sub-optimality, an empirical star algorithm (which we will refer to as STAR from now on) was then proposed in [1]; it was shown that the algorithm gives $O(\ln M/n)$ deviation bound with large probability under the flat prior $q_i = 1/M$. One major issue of the STAR algorithm is that its average performance is often inferior to EWMA, as we can see from our empirical examples. Therefore although theoretically interesting, it is not an algorithm that can be regarded as a replacement of EWMA for practical purposes. Partly for this reason, a more recent study [7] re-examined the problem of improving EWMA, where different estimators were proposed in order to achieve optimal deviation for model averaging. However, the proposed algorithms are rather complex and difficult to implement. The purpose of this paper is to present a simple greedy model averaging (GMA) algorithm that gives the optimal $O(\ln M/n)$ deviation bound with large probability, and it can be applied with arbitrary prior $q_i$. Moreover, unlike STAR which has average performance inferior to EWMA, the average performance of GMA algorithm is generally superior to EWMA as we shall illustrate with examples. It also has some other advantages which we will discuss in more details later in the paper.

## 2   Greedy Model Averaging

This paper studies a new model averaging procedure presented in Algorithm 1. The procedure has $L$ stages, and each time adds an additional model $\boldsymbol{f}_{\hat{k}(\ell)}$ into the ensemble. It is based on a simple, but

important modification of a classical sequential greedy approximation procedure in the literature [4], which corresponds to setting $\mu^{(\ell)} = 0$, $\lambda = 0$ in Algorithm 1 with $\alpha^{(\ell)}$ optimized over $[0, 1]$. The STAR algorithm corresponds to the stage-2 estimator $\hat{\boldsymbol{f}}^{(2)}$ with the above mentioned classical greedy procedure of [4]. However, in order to prove the desired deviation bound, our analysis critically depends on the extra term $\mu^{(\ell)} \left\| \hat{\boldsymbol{f}}^{(\ell-1)} - \boldsymbol{f}_j \right\|_2^2$ which isn't present in the classical procedure (that is, our proof does not apply to the procedure of [4]). As we will see in Section 4, this extra term does have a positive impact under suitable conditions that correspond to Theorem 1 and Theorem 2 below, and thus this term is not only for theoretical interest, but also it leads to practical benefits under the right conditions.

Another difference between GMA and the greedy algorithm in [4] is that our procedure allows the use of non-flat priors through the extra penalty term $\lambda c^{(\ell)} \ln(1/q_j)$. This generality can be useful for some applications. Moreover, it is useful to notice that if we choose the flat prior $q_j = 1/M$, then the term $\lambda c^{(\ell)} \ln(1/q_j)$ is identical for all models, and thus this term can be removed from the optimization. In this case, the proposed method has the advantage of being parameter free (with the default choice of $\nu = 0.5$). This advantage is also shared by the STAR algorithm.

---

**input**&emsp;&emsp;&emsp;: noisy observation $\boldsymbol{y}$ and static models $\boldsymbol{f}_1, \ldots, \boldsymbol{f}_M$
**output**&emsp;&emsp;: averaged model $\hat{\boldsymbol{f}}^{(\ell)}$
**parameters**: prior $\{q_j\}_{j=1,\ldots,M}$ and regularization parameters $\nu$ and $\lambda$

let $\hat{\boldsymbol{f}}^{(0)} = 0$
**for** $\ell = 1, 2, \ldots, L$ **do**
&emsp;let $\alpha^{(\ell)} = (\ell - 1)/\ell$
&emsp;let $\mu^{(1)} = 0$; $\mu^{(2)} = 0.05$; $\mu^{(\ell)} = \nu(\ell-1)/\ell^2$ if $\ell > 2$
&emsp;let $c^{(1)} = 1$; $c^{(2)} = 0.25$; and $c^{(\ell)} = [20\nu(1-\nu)(\ell-1)]^{-1}$ if $\ell > 2$
&emsp;let $\hat{k}^{(\ell)} = \operatorname{argmin}_j Q^{(\ell)}(j)$, where
$$Q^{(\ell)}(j) := \left[ \left\| \alpha^{(\ell)} \hat{\boldsymbol{f}}^{(\ell-1)} + (1 - \alpha^{(\ell)}) \boldsymbol{f}_j - \boldsymbol{y} \right\|_2^2 + \mu^{(\ell)} \left\| \hat{\boldsymbol{f}}^{(\ell-1)} - \boldsymbol{f}_j \right\|_2^2 + \lambda c^{(\ell)} \ln \frac{1}{q_j} \right]$$
&emsp;let $\hat{\boldsymbol{f}}^{(\ell)} = \alpha^{(\ell)} \hat{\boldsymbol{f}}^{(\ell-1)} + (1 - \alpha^{(\ell)}) \boldsymbol{f}_{\hat{k}^{(\ell)}}$
**end**

**Algorithm 1:** Greedy Model Averaging (GMA)

---

Observe that the first stage of GMA corresponds to the standard model selection procedure:

$$\hat{k}^{(1)} = \operatorname*{argmin}_j \left[ \left\| \boldsymbol{f}_j - \boldsymbol{y} \right\|_2^2 + \lambda \ln(1/q_j) \right],$$

$$\hat{\boldsymbol{f}}^{(1)} = \boldsymbol{f}_{\hat{k}^{(1)}}.$$

As we have pointed out earlier, it is well known that only $O(1/\sqrt{n})$ regret can be achieved by any model selection procedure (that is, any procedure that returns a single model $\hat{\boldsymbol{f}}_{\hat{k}}$ for some $\hat{k}$). However, a combination of only two models will allow us to achieve the optimal $O(1/n)$ rate. In fact, $\hat{\boldsymbol{f}}^{(2)}$ achieves this rate. For clarity, we rewrite this stage 2 estimator as

$$\hat{k}^{(2)} = \operatorname*{argmin}_j \left[ \left\| \frac{1}{2}(\boldsymbol{f}_{\hat{k}^{(1)}} + \boldsymbol{f}_j) - \boldsymbol{y} \right\|_2^2 + \frac{1}{20} \left\| \hat{\boldsymbol{f}}_{\hat{k}^{(1)}} - \boldsymbol{f}_j \right\|_2^2 + \frac{\lambda}{4} \ln(1/q_j) \right],$$

$$\hat{\boldsymbol{f}}^{(2)} = \frac{1}{2}(\boldsymbol{f}_{\hat{k}^{(1)}} + \boldsymbol{f}_{\hat{k}^{(2)}}).$$

Theorem 1 shows that this simple stage 2 estimator achieves $O(1/n)$ regret. A similar result was shown in [1] for the STAR algorithm under the flat prior $q_j = 1/M$, which corresponds to the stage 2 estimator of the classical greedy algorithm in [4]. Theoretically our result has several advantages over that of the classical EWMA method. First it produces a sparse estimator while exponential averaging estimator is dense; second the performance bound is scale free in the sense that the bound

depends only on the noise variance but not the magnitude of $\max_j \|\boldsymbol{f}_j\|$; third the optimal bound holds with high probability while EWMA only achieves optimal bound on average but not with large probability; and finally if we choose a flat prior $q_j = 1/M$, the estimator is parameter free because we can exclude the term $\lambda \ln(1/q_j)$ from the estimators. This result also improves the recent work of [7] in that the resulting bound is scale free while the algorithm itself is significantly simpler. One disadvantage of this stage-2 estimator (and similarly the STAR estimator of [1]) is that its average performance is generally inferior to that of EWMA, mainly due to the relatively large constant in Theorem 1 (the same issue holds for the STAR algorithm). For this reason, the stage-2 estimator is not a practical replacement of EWMA. This is the main reason why it is necessary to run GMA for $L > 2$ stages, which leads to reduced constants (see Theorem 2) below. Our empirical experiments show that in order to compete with EWMA for average performance, it is important to take $L > 2$. However a relatively small $L$ (as small as $L = 5$) is often sufficient, and in such case the resulting estimator is still quite sparse.

**Theorem 1** *Given* $q_j \geq 0$ *such that* $\sum_{j=1}^{M} q_j = 1$. *If* $\lambda \geq 40\sigma^2$, *then with probability* $1 - 2\delta$ *we have*

$$R(\hat{\boldsymbol{f}}^{(2)}) \leq \frac{\lambda}{n} \left[ \frac{3}{4} \ln(1/q_{k_*}) + \frac{1}{2} \ln(1/\delta) \right].$$

While the stage-2 estimator $\hat{\boldsymbol{f}}^{(2)}$ achieves the optimal rate, running GMA for more more stages can further improve the performance. The following theorem shows that similar bounds can be obtained for GMA at stages larger than 2. However, the constant before $\frac{\sigma^2}{n} \ln \frac{1}{q_{k_*}\delta}$ approaches 8 when $\ell \to \infty$ (with default $\nu = 0.5$), which is smaller than the constant of Theorem 1 which is about 30. This implies potential improvement when we run more stages, and this improvement is confirmed in our empirical study. In fact, with relatively large $\ell$, the GMA method not only has the theoretical advantage of achieving smaller regret in deviation (that is, the regret bound holds with large probability) but also achieves better average performance in practice.

**Theorem 2** *Given* $q_j \geq 0$ *such that* $\sum_{j=1}^{M} q_j = 1$. *If* $\lambda \geq 40\sigma^2$ *and let* $0 < \nu < 1$ *in Algorithm 1, then with probability* $1 - 2\delta$ *we have*

$$R(\hat{\boldsymbol{f}}^{(\ell)}) \leq \frac{\lambda}{n} \left[ \frac{(\ell - 2) + \ln(\ell - 1) + 30\nu(1 - \nu)}{20\nu(1 - \nu)\ell} \right] \ln \frac{1}{q_{k_*}\delta}.$$

Another important advantage of running GMA for $\ell > 2$ stages is that the resulting estimator not only competes with the best single estimator $\boldsymbol{f}_{k_*}$, but also competes with the best estimator in the convex hull of $\mathrm{cov}(\mathcal{F})$ (with the parameter $\nu$ appropriately tuned). Note that the latter can be significantly better than the former. Define the convex hull of $\mathcal{F}$ as

$$\mathrm{cov}(\mathcal{F}) = \left\{ \sum_{j=1}^{M} w_j \boldsymbol{f}_j : w_j \geq 0; \sum_j w_j = 1 \right\}.$$

The following theorem shows that as $\ell \to \infty$, the prediction error of $\hat{\boldsymbol{f}}^{(\ell)}$ is no more than $O(1/\sqrt{n})$ worse than that of the optimal $\bar{\boldsymbol{f}} \in \mathrm{cov}(\mathcal{F})$ when we choose a sufficiently small $\nu = O(1/\sqrt{n})$ in Algorithm 1. Note that in this case, it is beneficial to use a parameter $\nu$ smaller than the default choice of $\nu = 0.5$. This phenomenon is also confirmed by our experiments.

**Theorem 3** *Given* $q_j \geq 0$ *such that* $\sum_{j=1}^{M} q_j = 1$. *Consider any* $\{w_j : j = 1, \ldots, M\}$ *such that* $\sum_j w_j = 1$ *and* $w_j \geq 0$, *and let* $\bar{\boldsymbol{f}} = \sum_j w_j \boldsymbol{f}_j$. *If* $\lambda \geq 40\sigma^2$ *and let* $0 < \nu < 1$ *in Algorithm 1, then with probability* $1 - 2\delta$, *when* $\ell \to \infty$:

$$\frac{1}{n} \left\| \hat{\boldsymbol{f}}^{(\ell)} - \boldsymbol{g} \right\|_2^2 \leq \frac{1}{n} \left\| \bar{\boldsymbol{f}} - \boldsymbol{g} \right\|_2^2 + \frac{\nu}{n} \sum_k w_k \left\| \boldsymbol{f}_k - \bar{\boldsymbol{f}} \right\|_2^2 + \frac{\lambda}{20\nu(1 - \nu)n} \sum_k w_k \ln \left( \frac{1}{\delta q_k} \right) + O\left( \frac{1}{\ell} \right).$$

# 3 Experiments

The point of these experiments is to show that the consequences of our theoretical analysis can be observed in practice, which support the main conclusions we reach. For this purpose, we consider the model $\boldsymbol{g} = \boldsymbol{Xw} + 0.5\Delta\boldsymbol{g}$, where $\boldsymbol{X} = (\boldsymbol{f}_1, \ldots, \boldsymbol{f}_M)$ is an $n \times M$ matrix with independent standard Gaussian entries, and $\Delta\boldsymbol{g} \sim N(0, I_{n\times n})$ implies that the model is mis-specified.

The noise vector is $\boldsymbol{\xi} \sim N(0, \sigma^2 \boldsymbol{I}_{n\times n})$, independently generated of $\boldsymbol{X}$. The coefficient vector $\boldsymbol{w} = (w_1, \ldots, w_M)^\top$ is given by $w_i = |u_i| / \sum_{j=1}^s |u_j|$ for $i = 1, \ldots, s$, where $u_1, \ldots, u_s$ are independent standard uniform random variables for some fixed $s$.

The performance of an estimator $\hat{\boldsymbol{f}}$ measured here is the mean squared error (MSE) defined as

$$\mathrm{MSE}(\hat{\boldsymbol{f}}) = \frac{1}{n} \left\| \hat{\boldsymbol{f}} - \boldsymbol{g} \right\|_2^2.$$

We run the Greedy Model Averaging (GMA) algorithm for $L$ stages up to $L = 40$. The EWMA parameter is tuned via 10-fold cross-validation. Moreover, we also listed the performance of EWMA with projection, which is the method that runs EWMA, but with each model $\boldsymbol{f}_k$ replaced by model $\tilde{\boldsymbol{f}}_k = \alpha_k \boldsymbol{f}_k$ where $\alpha_k = \arg\min_{\alpha \in \mathbb{R}} \|\alpha \boldsymbol{f}_k - \boldsymbol{y}\|_2^2$. That is, $\tilde{\boldsymbol{f}}_k$ is the best linear scaling of $\boldsymbol{f}_k$ to predict $\boldsymbol{y}$. Note that this is a special case of the class of methods studied in [6] (which considers more general projections) that leads to non progressive regret bounds, and this is the method of significant current interests [3, 8]. However, at least for the scenario considered in our paper, the projected EWMA method never improves performance in our experiments. Finally, for reference purpose, we also report the MSE of the best single model (BSM) $\boldsymbol{f}_{k_*}$, where $k_*$ is given by (1). The model $\boldsymbol{f}_{k_*}$ is clearly not a valid estimator because it depends on the unobserved $\boldsymbol{g}$; however its performance is informative, and thus included in the tables. For simplicity, all algorithms use flat prior $q_k = 1/M$.

# 4 Illustration of Theorem 1 and Theorem 2

The first set of experiments are performed with the parameters $n = 50$, $M = 200$, $s = 1$ and $\sigma = 2$. Five hundred replications are run, and the MSE performance of different algorithms are reported in Table 1 using the "mean $\pm$ standard deviation" format.

Note that with $s = 1$, the target is $\boldsymbol{g} = \boldsymbol{f}_1 + 0.5\Delta\boldsymbol{g}$. Since $\boldsymbol{f}_1$ and $\Delta\boldsymbol{g}$ are random Gaussian vectors, the best single model is likely $\boldsymbol{f}_1$. The noise $\sigma = 2$ is relatively large. This is thus the situation that model averaging does not achieve as good a performance as that of the best single model. This corresponds to the scenario considered in Theorem 1 and Theorem 2.

The results indicate that for GMA, from $L = 1$ (corresponding to model selection) to $L = 2$ (stage-2 model averaging of Theorem 1), there is significant reduction of error. The performance of GMA with $L = 2$ is comparable to that of the STAR algorithm. This isn't surprising, because STAR can be regarded as the stage-2 estimator based on the more classical greedy algorithm of [4]. We also observe that the error keeps decreasing (but at a slower pace) when $L > 2$, which is consistent with Theorem 2. It means that in order to achieve good performance, it is necessary to use more stages than $L = 2$ (although this doesn't change the $O(1/n)$ rate for regret, it can significantly reduce constant). It becomes better than EWMA when $L$ is as small as $5$, which still gives a relatively sparse averaged model. EWMA with projection does not perform as well as the standard EWMA method in this setting. Moreover, we note that in this scenario, the standard choice of $\nu = 0.5$ in Theorem 2 is superior to choosing smaller $\nu = 0.1$ or $\nu = 0.001$. This again is consistent with Theorem 2, which shows that the new term we added into the greedy algorithm is indeed useful in this scenario.

# 5 Illustration of Theorem 3

The second set of experiments are performed with the parameters $n = 50$, $M = 200$, $s = 10$ and $\sigma = 0.5$. Five hundred replications are run, and the MSE performance of different algorithms are reported in Table 2 using the "mean $\pm$ standard deviation" format.

Table 1: MSE of different algorithms: best single model is superior to averaged models

| | STAR | EWMA | EWMA (with projection) | BSM |
|---|---|---|---|---|
| | $0.663 \pm 0.4$ | $0.645 \pm 0.5$ | $0.744 \pm 0.5$ | $0.252 \pm 0.05$ |

| GMA | $L = 1$ | $L = 2$ | $L = 5$ | $L = 20$ | $L = 40$ |
|---|---|---|---|---|---|
| $\nu = 0.5$ | $0.735 \pm 0.74$ | $0.689 \pm 0.4$ | $0.58 \pm 0.39$ | $0.566 \pm 0.37$ | $0.567 \pm 0.38$ |
| $\nu = 0.1$ | $0.735 \pm 0.74$ | $0.689 \pm 0.4$ | $0.645 \pm 0.31$ | $0.623 \pm 0.29$ | $0.622 \pm 0.29$ |
| $\nu = 0.01$ | $0.735 \pm 0.74$ | $0.689 \pm 0.4$ | $0.663 \pm 0.3$ | $0.638 \pm 0.28$ | $0.639 \pm 0.28$ |

Note that with $s = 10$, the target is $\boldsymbol{g} = \bar{\boldsymbol{f}} + 0.5\Delta\boldsymbol{g}$ for some $\bar{\boldsymbol{f}} \in \mathrm{cov}(\mathcal{F})$. The noise $\sigma = 0.5$ is relatively small, which makes it beneficial to compete with the best model $\bar{\boldsymbol{f}}$ in the convex hull even though GMA has a larger regret of $O(1/\sqrt{n})$ when competing with $\bar{\boldsymbol{f}}$. This is thus the situation considered in Theorem 3, which means that model averaging can achieve better performance than that of the best single model.

The results again show that for GMA, from $L = 1$ (corresponding to model selection) to $L = 2$ (stage-2 model averaging of Theorem 1), there is significant reduction of error. The performance of GMA with $L = 2$ is again comparable to that of the STAR algorithm. Again we observe that even with the standard choice of $\nu = 0.5$, the error keeps decreasing (but at a slower pace) when $L > 2$, which is consistent with Theorem 2. It becomes better than EWMA when $L$ is as small as $5$, which still gives a relatively sparse averaged model. EWMA with projection again does not perform as well as the standard EWMA method in this setting. Moreover, we note that in this scenario, the standard choice of $\nu = 0.5$ in Theorem 2 is inferior to choosing smaller parameter values of $\nu = 0.1$ or $\nu = 0.001$. This is consistent with Theorem 3, where it is beneficial to use a smaller value for $\nu$ in order to compete with the best model in the convex hull.

Table 2: MSE of different algorithms: best single model is inferior to averaged model

| | STAR | EWMA | EWMA (with projection) | BSM |
|---|---|---|---|---|
| | $0.443 \pm 0.08$ | $0.316 \pm 0.087$ | $0.364 \pm 0.078$ | $0.736 \pm 0.083$ |

| GMA | $L = 1$ | $L = 2$ | $L = 5$ | $L = 20$ | $L = 40$ |
|---|---|---|---|---|---|
| $\nu = 0.5$ | $0.809 \pm 0.12$ | $0.456 \pm 0.081$ | $0.305 \pm 0.062$ | $0.266 \pm 0.057$ | $0.265 \pm 0.057$ |
| $\nu = 0.1$ | $0.809 \pm 0.12$ | $0.456 \pm 0.081$ | $0.269 \pm 0.056$ | $0.214 \pm 0.046$ | $0.211 \pm 0.045$ |
| $\nu = 0.01$ | $0.809 \pm 0.12$ | $0.456 \pm 0.081$ | $0.268 \pm 0.053$ | $0.211 \pm 0.045$ | $0.207 \pm 0.045$ |

## 6  Conclusion

This paper presents a new model averaging scheme which we call greedy model averaging (GMA). It is shown that the new method can achieve regret bound of $O(\ln M/n)$ with large probability when competing with the single best model. Moreover, it can also compete with the best combined model in convex hull. Both our theory and experimental results suggest that the proposed GMA algorithm is superior to the standard EWMA procedure. Due to the simplicity of our proposal, GMA may be regarded as a valid alternative to the more widely studied EWMA procedure both for practical applications and for theoretical purposes. Finally we shall point out that while this work only considers static model averaging where the models $\mathcal{F}$ are finite, similar results can be obtained for affine estimators or infinite models considered in recent work [3, 6, 8]. Such extension will be left to the extended report.

## A  Proof Sketches

We only include proof sketches, and leave the details to the supplemental material that accompanies the submission. First we need the following standard Gaussian tail bounds. The proofs can be found in the supplemental material.

**Proposition 1** *Let $\boldsymbol{f}_j \in \mathbb{R}^n$ be a set of fixed vectors ($j = 1, \ldots, M$), and assume that $q_j \geq 0$ with $\sum_j q_j = 1$. Let $k_*$ be a fixed integer between $1$ and $M$. Define event $E_1$ as*

$$E_1 = \left\{ \forall j : (\boldsymbol{f}_j - \boldsymbol{f}_{k_*})^\top \boldsymbol{\xi} \leq \sigma \|\boldsymbol{f}_j - \boldsymbol{f}_{k_*}\|_2 \sqrt{2\ln(1/(\delta q_j))} \right\}$$

*and define event $E_2$ as*

$$E_2 = \left\{ \forall j, k : (\boldsymbol{f}_j - \boldsymbol{f}_k)^\top \boldsymbol{\xi} \leq \sigma \|\boldsymbol{f}_j - \boldsymbol{f}_k\|_2 \sqrt{2\ln(1/(\delta q_j q_k))} \right\},$$

*then $P(E_1) \geq 1 - \delta$ and $P(E_2) \geq 1 - \delta$.*

### A.1 Proof Sketch of Theorem 1

More detailed proof can be found in the supplemental material. Note that with probability $1 - 2\delta$, both event $E_1$ and event $E_2$ of Proposition 1 hold. Moreover we have

$$
\begin{aligned}
\left\| \hat{\boldsymbol{f}}^{(2)} - \boldsymbol{g} \right\|_2^2 
&= \left\| \alpha^{(2)} \hat{\boldsymbol{f}}^{(1)} + (1 - \alpha^{(2)}) \boldsymbol{f}_{\hat{k}^{(2)}} - \boldsymbol{g} \right\|_2^2 \\
&\leq \left\| \alpha^{(2)} \hat{\boldsymbol{f}}^{(1)} + (1 - \alpha^{(2)}) \boldsymbol{f}_{k_*} - \boldsymbol{g} \right\|_2^2 + 2(1 - \alpha^{(2)}) \boldsymbol{\xi}^\top (\boldsymbol{f}_{\hat{k}^{(2)}} - \boldsymbol{f}_{k_*}) \\
&\quad + \mu^{(2)} \left( \left\| \hat{\boldsymbol{f}}^{(1)} - \boldsymbol{f}_{k_*} \right\|_2^2 - \left\| \hat{\boldsymbol{f}}^{(1)} - \boldsymbol{f}_{\hat{k}^{(2)}} \right\|_2^2 \right) + \lambda c^{(2)} (\ln(1/q_{k_*}) - \ln(1/q_{\hat{k}^{(2)}})).
\end{aligned}
$$

In the above derivation, the inequality is equivalent to $Q^{(2)}(\hat{k}^{(2)}) \leq Q^{(2)}(k_*)$, which is a simple fact of the definition of $\hat{k}^{(\ell)}$ in the algorithm. Also we can rewrite the fact that $Q^{(1)}(\hat{k}^{(1)}) \leq Q^{(1)}(k_*)$ as

$$\left\| \hat{\boldsymbol{f}}^{(1)} - \boldsymbol{g} \right\|_2^2 - \left\| \boldsymbol{f}_{k_*} - \boldsymbol{g} \right\|_2^2 \leq 2\boldsymbol{\xi}^\top (\boldsymbol{f}_{\hat{k}^{(1)}} - \boldsymbol{f}_{k_*}) + \lambda c^{(1)} \ln(q_{\hat{k}^{(1)}}/q_{k_*}).$$

By combining the above two inequalities, we obtain

$$
\begin{aligned}
\left\| \hat{\boldsymbol{f}}^{(2)} - \boldsymbol{g} \right\|_2^2 - \left\| \boldsymbol{f}_{k_*} - \boldsymbol{g} \right\|_2^2 &\leq \alpha^{(2)} \left[ 2\boldsymbol{\xi}^\top (\boldsymbol{f}_{\hat{k}^{(1)}} - \boldsymbol{f}_{k_*}) + \lambda c^{(1)} \ln(q_{\hat{k}^{(1)}}/q_{k_*}) \right] \\
&\quad + 2(1 - \alpha^{(2)}) \boldsymbol{\xi}^\top (\boldsymbol{f}_{\hat{k}^{(2)}} - \boldsymbol{f}_{k_*}) + \left[ \mu^{(2)} - \alpha^{(2)}(1 - \alpha^{(2)}) \right] \left\| \boldsymbol{f}_{\hat{k}^{(1)}} - \boldsymbol{f}_{k_*} \right\|_2^2 \\
&\quad - \mu^{(2)} \left\| \boldsymbol{f}_{\hat{k}^{(1)}} - \boldsymbol{f}_{\hat{k}^{(2)}} \right\|_2^2 + \lambda c^{(2)} (\ln(1/q_{k_*}) - \ln(1/q_{\hat{k}^{(2)}})).
\end{aligned}
$$

Since $\alpha^{(2)} = 1/2$, we obtain

$$
\begin{aligned}
&\left\| \hat{\boldsymbol{f}}^{(2)} - \boldsymbol{g} \right\|_2^2 - \left\| \boldsymbol{f}_{k_*} - \boldsymbol{g} \right\|_2^2 \\
&\leq (\tfrac{1}{2}\lambda c^{(1)} + \lambda c^{(2)}) \ln(1/q_{k_*}) - \tfrac{1}{2}\lambda c^{(1)} \ln(1/q_{\hat{k}^{(1)}}) - \lambda c^{(2)} \ln(1/q_{\hat{k}^{(2)}}) \\
&\quad + 2 \left\| \boldsymbol{f}_{\hat{k}^{(1)}} - \boldsymbol{f}_{k_*} \right\|_2 \sigma \sqrt{2\ln \frac{1}{q_{\hat{k}^{(1)}} \delta}} + 2 \cdot \tfrac{1}{2} \left\| \boldsymbol{f}_{\hat{k}^{(2)}} - \boldsymbol{f}_{\hat{k}^{(1)}} \right\|_2 \sigma \sqrt{2\ln \frac{1}{q_{\hat{k}^{(1)}} q_{\hat{k}^{(2)}} \delta}} \\
&\quad + (\mu^{(2)} - 1/4) \left\| \boldsymbol{f}_{\hat{k}^{(1)}} - \boldsymbol{f}_{k_*} \right\|_2^2 - \mu^{(2)} \left\| \boldsymbol{f}_{\hat{k}^{(1)}} - \boldsymbol{f}_{\hat{k}^{(2)}} \right\|_2^2 \\
&\leq (\tfrac{1}{2}\lambda c^{(1)} + \lambda c^{(2)}) \ln(1/q_{k_*}) + (2r_1 + 2r_2) \ln(1/\delta).
\end{aligned}
$$

The first inequality above uses the tail probability bounds in the event $E_1$ and $E_2$. We then use the algebraic inequality $2a_1 b_1 \leq a_1^2/r_1 + r_1 b_1^2$ and $2a_2 b_2 \leq a_2^2/r_2 + r_2 b_2^2$ to obtain the last inequality, which implies the desired bound.

### A.2 Proof Sketch of Theorem 2

Again, more detailed proof can be found in the supplemental material. With probability $1 - 2\delta$, both event $E_1$ and event $E_2$ of Proposition 1 hold. This implies that the claim of Theorem 1 also holds.

Now consider any $\ell \geq 3$. We have

$$\left\| \hat{\boldsymbol{f}}^{(\ell)} - \boldsymbol{g} \right\|_2^2 \leq \left\| \alpha^{(\ell)} \hat{\boldsymbol{f}}^{(\ell-1)} + (1 - \alpha^{(\ell)}) \boldsymbol{f}_{k_*} - \boldsymbol{g} \right\|_2^2 + 2\boldsymbol{\xi}^\top \left[ (1 - \alpha^{(\ell)}) \boldsymbol{f}_{\hat{k}^{(\ell)}} - (1 - \alpha^{(\ell)}) \boldsymbol{f}_{k_*} \right]$$
$$+ \lambda c^{(\ell)} (\ln(1/q_{k_*}) - \ln(1/q_{\hat{k}^{(\ell)}})) + \mu^{(\ell)} \left( \left\| \hat{\boldsymbol{f}}^{(\ell-1)} - \boldsymbol{f}_{k_*} \right\|_2^2 - \left\| \hat{\boldsymbol{f}}^{(\ell-1)} - \boldsymbol{f}_{\hat{k}^{(\ell)}} \right\|_2^2 \right).$$

The inequality is equivalent to $Q^{(\ell)}(\hat{k}^{(\ell)}) \leq Q^{(\ell)}(k_*)$, which is a simple fact of the definition of $\hat{k}^{(\ell)}$ in the algorithm. We can rewrite the above inequality as

$$\left( \left\| \hat{\boldsymbol{f}}^{(\ell)} - \boldsymbol{g} \right\|_2^2 - \left\| \boldsymbol{f}_{k_*} - \boldsymbol{g} \right\|_2^2 \right)$$

$$\leq \alpha^{(\ell)} \left( \left\| \hat{\boldsymbol{f}}^{(\ell-1)} - \boldsymbol{g} \right\|_2^2 - \left\| \boldsymbol{f}_{k_*} - \boldsymbol{g} \right\|_2^2 \right) - \lambda c^{(\ell)} (\ln(q_{k_*}) - \ln(q_{\hat{k}^{(\ell)}})) + 2(1 - \alpha^{(\ell)}) \boldsymbol{\xi}^\top (\boldsymbol{f}_{\hat{k}^{(\ell)}} - \boldsymbol{f}_{k_*})$$

$$- \mu^{(\ell)} \left\| \boldsymbol{f}_{\hat{k}^{(\ell)}} - \hat{\boldsymbol{f}}^{(\ell-1)} \right\|_2^2 + \left[ \mu^{(\ell)} - \alpha^{(\ell)} (1 - \alpha^{(\ell)}) \right] \left\| \hat{\boldsymbol{f}}^{(\ell-1)} - \boldsymbol{f}_{k_*} \right\|_2^2$$

$$\leq \alpha^{(\ell)} \left( \left\| \hat{\boldsymbol{f}}^{(\ell-1)} - \boldsymbol{g} \right\|_2^2 - \left\| \boldsymbol{f}_{k_*} - \boldsymbol{g} \right\|_2^2 \right) + \lambda c^{(\ell)} (\ln(1/q_{k_*}) - \ln(1/q_{\hat{k}^{(\ell)}}))$$

$$+ \frac{2}{\ell} \left\| \boldsymbol{f}_{\hat{k}^{(\ell)}} - \boldsymbol{f}_{k_*} \right\|_2 \sigma \sqrt{2 \ln \frac{1}{q_{\hat{k}^{(\ell)}} \delta}} - \frac{\mu^{(\ell)} \left[ \alpha^{(\ell)} (1 - \alpha^{(\ell)}) - \mu^{(\ell)} \right]}{\alpha^{(\ell)} (1 - \alpha^{(\ell)})} \left\| \boldsymbol{f}_{\hat{k}^{(\ell)}} - \boldsymbol{f}_{k_*} \right\|_2^2$$

$$\leq \frac{\ell - 1}{\ell} \left( \left\| \hat{\boldsymbol{f}}^{(\ell-1)} - \boldsymbol{g} \right\|_2^2 - \left\| \boldsymbol{f}_{k_*} - \boldsymbol{g} \right\|_2^2 \right) + \lambda c^{(\ell)} (\ln(1/q_{k_*}) - \ln(1/q_{\hat{k}^{(\ell)}}))$$

$$+ \left[ -\frac{\ell - 1}{\ell^2} \nu(1 - \nu) + \frac{\sigma^2}{\ell^2 r_\ell} \right] \left\| \boldsymbol{f}_{\hat{k}^{(\ell)}} - \boldsymbol{f}_{k_*} \right\|_2^2 + 2r_\ell \ln \frac{1}{q_{\hat{k}^{(\ell)}} \delta}.$$

The second inequality uses the fact that $-p \|a\|^2 - q \|b\|^2 \leq -pq/(p + q) \|a + b\|^2$, which implies that $[\mu^{(\ell)} - \alpha^{(\ell)} (1 - \alpha^{(\ell)})] \left\| \hat{\boldsymbol{f}}^{(\ell-1)} - \boldsymbol{f}_{k_*} \right\|_2^2 - \mu^{(\ell)} \left\| \boldsymbol{f}_{\hat{k}^{(\ell)}} - \hat{\boldsymbol{f}}^{(\ell-1)} \right\|_2^2 \leq -\frac{\mu^{(\ell)} [\alpha^{(\ell)} (1 - \alpha^{(\ell)}) - \mu^{(\ell)}]}{\alpha^{(\ell)} (1 - \alpha^{(\ell)})} \left\| \boldsymbol{f}_{\hat{k}^{(\ell)}} - \boldsymbol{f}_{k_*} \right\|_2^2$ and uses the Gaussian tail bound in the event $E_1$. The last inequality uses $2ab \leq a^2/r_\ell + r_\ell b^2$, where $r_\ell > 0$ is $r_\ell = \lambda c^{(\ell)}/2$. Denote by $R^{(\ell)} = \left\| \hat{\boldsymbol{f}}^{(\ell)} - \boldsymbol{g} \right\|_2^2 - \left\| \boldsymbol{f}_{k_*} - \boldsymbol{g} \right\|_2^2$, then since the choice of parameters $c^{(\ell)} = [20\nu(1 - \nu)(\ell - 1)]^{-1}$ we obtain $R^{(\ell)} \leq \frac{\ell-1}{\ell} R^{(\ell-1)} + \lambda c^{(\ell)} \ln(1/q_{k_*} \delta)$. Solving this recursion for $R^{(\ell)}$ leads to the desired bound.

### A.3   Proof Sketch of Theorem 3

Again, more detailed proof can be found in the supplemental material. Consider any $\ell \geq 3$. We have

$$\left\| \hat{\boldsymbol{f}}^{(\ell)} - \boldsymbol{g} \right\|_2^2 \leq$$

$$\sum_k w_k \left\| \alpha^{(\ell)} \hat{\boldsymbol{f}}^{(\ell-1)} + (1 - \alpha^{(\ell)}) \boldsymbol{f}_k - \boldsymbol{g} \right\|_2^2 + \mu^{(\ell)} \left( \sum_k w_k \left\| \hat{\boldsymbol{f}}^{(\ell-1)} - \boldsymbol{f}_k \right\|_2^2 - \left\| \hat{\boldsymbol{f}}^{(\ell-1)} - \boldsymbol{f}_{\hat{k}^{(\ell)}} \right\|_2^2 \right)$$

$$+ \lambda c^{(\ell)} (\sum_k w_k \ln(1/q_k) - \ln(1/q_{\hat{k}^{(\ell)}})) + 2\boldsymbol{\xi}^\top \left[ (1 - \alpha^{(\ell)}) \boldsymbol{f}_{\hat{k}^{(\ell)}} - (1 - \alpha^{(\ell)}) \sum_k w_k \boldsymbol{f}_k \right].$$

The inequality is equivalent to $Q^{(\ell)}(\hat{k}^{(\ell)}) \leq \sum_k w_k Q^{(\ell)}(k)$, which is a simple fact of the definition of $\hat{k}^{(\ell)}$ in the algorithm. Denote by $R^{(\ell)} = \left\| \hat{\boldsymbol{f}}^{(\ell)} - \boldsymbol{g} \right\|_2^2 - \left\| \bar{\boldsymbol{f}} - \boldsymbol{g} \right\|_2^2$, then the same derivation as that of Theorem 2 implies that

$$R^{(\ell)} \leq \frac{\ell - 1}{\ell} R^{(\ell-1)} + \lambda c^{(\ell)} \sum_k w_k \ln(1/(\delta q_k)) + [\mu^{(\ell)} + (1 - \alpha^{(\ell)})^2] \sum_k w_k \left\| \boldsymbol{f}_k - \bar{\boldsymbol{f}} \right\|_2^2.$$

Now by solving the recursion, we obtain the theorem.

# References

[1] Jean-Yves Audibert. Progressive mixture rules are deviation suboptimal. In *NIPS'07*, 2008.

[2] Olivier Catoni. *Statistical learning theory and stochastic optimization*. Springer-Verlag, 2004.

[3] Arnak Dalalyan and Joseph Salmon. Optimal aggregation of affine estimators. In *COLT'01*, 2011.

[4] L.K. Jones. A simple lemma on greedy approximation in Hilbert space and convergence rates for projection pursuit regression and neural network training. *Ann. Statist.*, 20(1):608–613, 1992.

[5] Anatoli Juditsky, Philippe Rigollet, and Alexandre Tsybakov. Learning by mirror averaging. *The Annals of Statistics*, 36:2183–2206, 2008.

[6] Gilbert Leung and A.R. Barron. Information theory and mixing least-squares regressions. *Information Theory, IEEE Transactions on*, 52(8):3396 –3410, aug. 2006.

[7] Philippe Rigollet. Kullback-leibler aggregation and misspecified generalized linear models. arXiv:0911.2919, November 2010.

[8] Pilippe Rigollet and Alexandre Tsybakov. Exponential Screening and optimal rates of sparse estimation. *The Annals of Statistics*, 39:731–771, 2011.

[9] Yuhong Yang. Adaptive regression by mixing. *Journal of the American Statistical Association*, 96:574–588, 2001.

